# Robust Clustering as Ensembles of Affinity Relations

**Hairong Liu[1], Longin Jan Latecki[2], Shuicheng Yan[1]**
[1]Department of Electrical and Computer Engineering, National University of Singapore, Singapore
[2]Department of Computer and Information Sciences, Temple University, Philadelphia, USA
`lhrbss@gmail.com,latecki@temple.edu,eleyans@nus.edu.sg`

## Abstract

In this paper, we regard clustering as ensembles of $k$-ary affinity relations and clusters correspond to subsets of objects with maximal average affinity relations. The average affinity relation of a cluster is relaxed and well approximated by a constrained homogenous function. We present an efficient procedure to solve this optimization problem, and show that the underlying clusters can be robustly revealed by using priors systematically constructed from the data. Our method can automatically select some points to form clusters, leaving other points un-grouped; thus it is inherently robust to large numbers of outliers, which has seriously limited the applicability of classical methods. Our method also provides a unified solution to clustering from $k$-ary affinity relations with $k \geq 2$, that is, it applies to both graph-based and hypergraph-based clustering problems. Both theoretical analysis and experimental results show the superiority of our method over classical solutions to the clustering problem, especially when there exists a large number of outliers.

## 1 Introduction

Data clustering is a fundamental problem in many fields, such as machine learning, data mining and computer vision [1]. Unfortunately, there is no universally accepted definition of a cluster, probably because of the diverse forms of clusters in real applications. But it is generally agreed that the objects belonging to a cluster satisfy certain internal coherence condition, while the objects not belonging to a cluster usually do not satisfy this condition.

Most of existing clustering methods are partition-based, such as k-means [2], spectral clustering [3, 4, 5] and affinity propagation [6]. These methods implicitly share an assumption: every data point must belong to a cluster. This assumption greatly simplifies the problem, since we do not need to judge whether a data point is an outlier or not, which is very challenging. However, this assumption also results in bad performance of these methods when there exists a large number of outliers, as frequently met in many real-world applications.

The criteria to judge whether several objects belong to the same cluster or not are typically expressed by pairwise relations, which is encoded as the weights of an affinity graph. However, in many applications, high order relations are more appropriate, and may even be the only choice, which naturally results in hyperedges in hypergraphs. For example, when clustering a given set of points into lines, pairwise relations are not meaningful, since every pair of data points trivially defines a line. However, for every three data points, whether they are near collinear or not conveys very important information.

As graph-based clustering problem has been well studied, many researchers tried to deal with hypergraph-based clustering by using existing graph-based clustering methods. One direction is to transform a hypergraph into a graph, whose edge-weights are mapped from the weights of the original hypergraph. Zien et. al. [7] proposed two approaches called "clique expansion" and "star expansion", respectively, for such a purpose. Rodriguez [8] showed the relationship between the

spectral properties of the Laplacian matrix of the resulting graph and the minimum cut of the original hypergraph. Agarwal et al. [9] proposed the "clique averaging" method and reported better results than "clique expansion" method. Another direction is to generalize graph-based clustering method to hypergraphs. Zhou et al. [10] generalized the well-known "normalized cut" method [5] and defined a hypergraph normalized cut criterion for a $k$-partition of the vertices. Shashua et al. [11] cast the clustering problem with high order relations into a nonnegative factorization problem of the closest hyper-stochastic version of the input affinity tensor.

Based on game theory, Bulo and Pelillo [12] proposed to consider the hypergraph-based clustering problem as a multi-player non-cooperative "clustering game" and solve it by replicator equation, which is in fact a generalization of their previous work [13]. This new formulation has a solid theoretical foundation, possesses several appealing properties, and achieved state-of-art results. This method is in fact a specific case of our proposed method, and we will discuss this point in Section 2.

In this paper, we propose a unified method for clustering from $k$-ary affinity relations, which is applicable to both graph-based and hypergraph-based clustering problems. Our method is motivated by an intuitive observation: for a cluster with $m$ objects, there may exist $\binom{m}{k}$ possible $k$-ary affinity relations, and most of these (sometimes even all) $k$-ary affinity relations should agree with each other on the same criterion. For example, in the line clustering problem, for $m$ points on the same line, there are $\binom{m}{3}$ possible triplets, and all these triplets should satisfy the criterion that they lie on a line. The ensemble of such large number of affinity relations is hardly produced by outliers and is also very robust to noises, thus yielding a robust mechanism for clustering.

## 2 Formulation

Clustering from $k$-ary affinity relations can be intuitively described as clustering on a special kind of edge-weighted hypergraph, $k$-graph. Formally, a $k$-graph is a triplet $G = (V, E, w)$, where $V = \{1, \cdots, n\}$ is a finite set of vertices, with each vertex representing an object, $E \subseteq V^k$ is the set of hyperedges, with each hyperedge representing a $k$-ary affinity relation, and $w : E \rightarrow R$ is a weighting function which associates a real value (can be negative) with each hyperedge, with larger weights representing stronger affinity relations. We only consider the $k$-ary affinity relations with no duplicate objects, that is, the hyperedges among $k$ different vertices. For hyperedges with duplicated vertices, we simply set their weights to zeros.

Each hyperedge $e \in E$ involves $k$ vertices, thus can be represented as $k$-tuple $\{v_1, \cdots, v_k\}$. The weighted adjacency array of graph $G$ is an $\overbrace{n \times n \times \cdots \times n}^{k}$ super-symmetry array, denoted by $M$, and defined as

$$M(v_1, \cdots, v_k) = \begin{cases} w(\{v_1, \cdots, v_k\}) & \text{if } \{v_1, \cdots, v_k\} \in E, \\ 0 & \text{else,} \end{cases} \tag{1}$$

Note that each edge $\{v_1, \cdots, v_k\} \in E$ has $k!$ duplicate entries in the array $M$.

For a subset $U \subseteq V$ with $m$ vertices, its edge set is denoted as $E_U$. If $U$ is really a cluster, then most of hyperedges in $E_U$ should have large weights. The simplest measure to reflect such ensemble phenomenon is the sum of all entries in $M$ whose corresponding hyperedges contain only vertices in $U$, which can be expressed as:

$$S(U) = \sum_{v_1, \cdots, v_k \in U} M(v_1, \cdots, v_k). \tag{2}$$

Suppose $y$ is an $n \times 1$ indicator vector of the subset $U$, such that $y_{v_i} = 1$ if $v_i \in U$ and zero otherwise, then $S(U)$ can be expressed as:

$$S(U) = S(y) = \sum_{v_1, \cdots, v_k \in V} M(v_1, \cdots, v_k) \overbrace{y_{v_1} \cdots y_{v_k}}^{k}. \tag{3}$$

Obviously, $S(U)$ usually increases as the number of vertices in $U$ increases. Since $\sum_i y_i = m$ and there are $m^k$ summands in $S(U)$, the average of these entries can be expressed as:

$$S_{av}(U) = \frac{1}{m^k} S(y)$$

$$= \frac{1}{m^k} \sum_{v_1,\cdots,v_k \in V} M(v_1,\cdots,v_k) \overbrace{y_{v_1} \cdots y_{v_k}}^{k}$$

$$= \sum_{v_1,\cdots,v_k \in V} M(v_1,\cdots,v_k) \overbrace{\frac{y_{v_1}}{m} \cdots \frac{y_{v_k}}{m}}^{k}$$

$$= \sum_{v_1,\cdots,v_k \in V} M(v_1,\cdots,v_k) \overbrace{x_{v_1} \cdots x_{v_k}}^{k}, \tag{4}$$

where $x = y/m$. As $\sum_i y_i = m$, $\sum_i x_i = 1$ is a natural constraint over $x$.

Intuitively, when $U$ is a true cluster, $S_{av}(U)$ should be relatively large. Thus, the clustering problem corresponds to the problem of maximizing $S_{av}(U)$. In essence, this is a combinatorial optimization problem, since we know neither $m$ nor which $m$ objects to select. As this problem is NP-hard, to reduce its complexity, we relax $x$ to be within a continuous range $[0, \varepsilon]$, where $\varepsilon \leq 1$ is a constant, while keeping the constraint $\sum_i x_i = 1$. Then the problem becomes:

$$\begin{cases} \max f(x) = \sum_{v_1,\cdots,v_k \in V} M(v_1,\cdots,v_k) \prod_{i=1}^{k} x_{v_i}, \\ \text{subject to } x \in \Delta^n \text{ and } x_i \in [0,\varepsilon] \end{cases} \tag{5}$$

where $\Delta^n = \{x \in \mathrm{R}^n : x \geq 0 \text{ and } \sum_i x_i = 1\}$ is the standard simplex in $\mathrm{R}^n$. Note that $S_{av}(x)$ is abbreviated by $f(x)$ to simplify the formula.

The adoption of $\ell_1$-norm in (5) not only let $x_i$ have an intuitive probabilistic meaning, that is, $x_i$ represents the probability for the cluster contain the $i$-th object, but also makes the solution sparse, which means to automatically select some objects to form a cluster, while ignoring other objects.

**Relation to Clustering Game.** In [12], Bulo and Pelillo proposed to cast the hypergraph-based clustering problem into a *clustering game*, which leads to a similar formulation as (5). In fact, their formulation is a special case of (5) when $\varepsilon = 1$. Setting $\varepsilon < 1$ means that the probability of choosing each strategy (from game theory perspective) or choosing each object (from our perspective) has an known upper bound, which is in fact a prior, while $\varepsilon = 1$ represents a noninformative prior. This point is very essential in many applications, it avoids the phenomenon where some components of $x$ dominate. For example, if the weight of a hyperedge is extremely large, then the cluster may only select the vertices associated with this hyperedge, which is usually not desirable. In fact, $\varepsilon$ offers us a tool to control the least number of objects in cluster. Since each component does not exceed $\varepsilon$, the cluster contains at least $[\frac{1}{\varepsilon}]$ objects, where $[z]$ represents the smallest integer larger than or equal to $z$. Because of the constraint $x_i \in [0, \varepsilon]$, the solution is also totally different from [12].

## 3 Algorithm

Formulation (5) usually has many local maxima. Large maxima correspond to true clusters and small maxima usually form meaningless subsets. In this section, we first analyze the properties of the maximizer $x^*$, which are critical in algorithm design, and then introduce our algorithm to calculate $x^*$.

Since the formulation (5) is a constrained optimization problem, by adding Lagrangian multipliers $\lambda, \mu_1, \cdots, \mu_n$ and $\beta_1, \cdots, \beta_n$, $\mu_i \geq 0$ and $\beta_i \geq 0$ for all $i = 1, \cdots, n$, we can obtain its Lagrangian function:

$$L(x, \lambda, \mu, \beta) = f(x) - \lambda \left( \sum_{i=1}^{n} x_i - 1 \right) + \sum_{i=1}^{n} \mu_i x_i + \sum_{i=1}^{n} \beta_i (\varepsilon - x_i). \tag{6}$$

The *reward* at vertex $i$, denoted by $r_i(x)$, is defined as follows:

$$r_i(x) = \sum_{v_1,\cdots,v_{k-1} \in V} M(v_1,\cdots,v_{k-1},i) \prod_{t=1}^{k-1} x_{v_t} \tag{7}$$

Since $M$ is a super-symmetry array, then $\frac{\partial f(x)}{\partial x_i} = k r_i(x)$, i.e., $r_i(x)$ is proportional to the gradient of $f(x)$ at $x$.

Any local maximizer $x^*$ must satisfy the Karush-Kuhn-Tucker (KKT) condition [14], i.e., the first-order necessary conditions for local optimality. That is,

$$\begin{cases} kr_i(x^*) - \lambda + \mu_i - \beta_i = 0, \ i = 1, \cdots, n, \\ \sum_{i=1}^{n} x_i^* \mu_i = 0, \\ \sum_{i=1}^{n} (\varepsilon - x_i^*) \beta_i = 0. \end{cases} \tag{8}$$

Since $x_i^*$, $\mu_i$ and $\beta_i$ are all nonnegative for all $i$'s, $\sum_{i=1}^{n} x_i^* \mu_i = 0$ is equivalent to saying that if $x_i^* > 0$, then $\mu_i = 0$, and $\sum_{i=1}^{n} (\varepsilon - x_i^*) \beta_i = 0$ is equivalent to saying that if $x_i^* < \varepsilon$, then $\beta_i = 0$. Hence, the KKT conditions can be rewritten as:

$$r_i(x^*) \begin{cases} \leq \lambda/k, & x_i^* = 0, \\ = \lambda/k, & x_i^* > 0 \text{ and } x_i^* < \varepsilon, \\ \geq \lambda/k, & x_i^* = \varepsilon. \end{cases} \tag{9}$$

According to $x$, the vertices set $V$ can be divided into three disjoint subsets, $V_1(x) = \{i | x_i = 0\}$, $V_2(x) = \{i | x_i \in (0, \varepsilon)\}$ and $V_3(x) = \{i | x_i = \varepsilon\}$. The Equation (9) characterizes the properties of the solution of (5), which are further summarized in the following theorem.

**Theorem 1**. If $x^*$ is the solution of (5), then there exists a constant $\eta$ ($= \lambda/k$) such that 1) the rewards at all vertices belonging to $V_1(x^*)$ are not larger than $\eta$; 2) the rewards at all vertices belonging to $V_2(x^*)$ are equal to $\eta$; and 3) the rewards at all vertices belonging to $V_3(x^*)$ are not smaller than $\eta$.

**Proof**: Since KKT condition is a necessary condition, according to (9), the solution $x^*$ must satisfy 1), 2) and 3).

The set of non-zero components is $V_d(x) = V_2(x) \cup V_3(x)$ and the set of the components which are smaller than $\varepsilon$ is $V_u(x) = V_1(x) \cup V_2(x)$. For any $x$, if we want to update it to increase $f(x)$, then the values of some components belonging to $V_d(x)$ must decrease and the values of some components belonging to $V_u(x)$ must increase. According to Theorem 1, if $x$ is the solution of (5), then $r_i(x) \leq r_j(x), \forall i \in V_u(x), \forall j \in V_d(x)$. On the contrary, if $\exists i \in V_u(x), \exists j \in V_d(x), r_i(x) > r_j(x)$, then $x$ is not the solution of (5). In fact, in such case, we can increase $x_i$ and decrease $x_j$ to increase $f(x)$. That is, let

$$x_l' = \begin{cases} x_l, & l \neq i, l \neq j; \\ x_l + \alpha, & l = i; \\ x_l - \alpha, & l = j. \end{cases} \tag{10}$$

and define

$$r_{ij}(x) = \sum_{v_1, \cdots, v_{k-2}} M(v_1, \cdots, v_{k-2}, i, j) \prod_{t=1}^{k-2} x_{v_t} \tag{11}$$

Then

$$f(x') - f(x) = -k(k-1)r_{ij}(x)\alpha^2 + k(r_i(x) - r_j(x))\alpha \tag{12}$$

Since $r_i(x) > r_j(x)$, we can always select a proper $\alpha > 0$ to increase $f(x)$. According to formula (10) and the constraint over $x_i$, $\alpha \leq \min(x_j, \varepsilon - x_i)$. Since $r_i(x) > r_j(x)$, if $r_{ij}(x) \leq 0$, then when $\alpha = \min(x_j, \varepsilon - x_i)$, the increase of $f(x)$ reaches maximum; if $r_{ij} > 0$, then when $\alpha = \min(x_j, \varepsilon - x_i, \frac{r_i(x) - r_j(x)}{2(k-1)r_{ij}(x)})$, the increase of $f(x)$ reaches maximum.

According to the above analysis, if $\exists i \in V_u(x), \exists j \in V_d(x), r_i(x) > r_j(x)$, then we can update $x$ to increase $f(x)$. Such procedure iterates until $r_i(x) \leq r_j(x), \forall i \in V_u(x), \forall j \in V_d(x)$. From a prior (initialization) $x(0)$, the algorithm to compute the local maximizer of (5) is summarized in Algorithm 1, which successively chooses the "best" vertex and the "worst" vertex and then update their corresponding components of $x$.

Since significant maxima of formulation (5) usually correspond to true clusters, we need multiple initializations (priors) to obtain them, with at least one initialization at the basin of attraction of every significant maximum. Such informative priors in fact can be easily and efficiently constructed from the neighborhood of every vertex (vertices with hyperedges connecting to this vertex), because the neighbors of a vertex generally have much higher probabilities to belong to the same cluster.

---

**Algorithm 1** Compute a local maximizer $x^*$ from a prior $x(0)$

---

1: **Input:** Weighted adjacency array $M$, prior $x(0)$;
2: **repeat**
3:     Compute the reward $r_i(x)$ for each vertex $i$;
4:     Compute $V_1(x(t))$, $V_2(x(t))$, $V_3(x(t))$, $V_d(x(t))$, and $V_u(x(t))$;
5:     Find the vertex $i$ in $V_u(x(t))$ with the largest reward and the vertex $j$ in $V_d(x(t))$ with the smallest reward;
6:     Compute $\alpha$ and update $x(t)$ by formula (10) to obtain $x(t+1)$;
7: **until** $x$ is a local maximizer
8: **Output:** The local maximizer $x^*$.

---

---

**Algorithm 2** Construct a prior $x(0)$ containing vertex $v$

---

1: **Input:** Hyperedge set $E(v)$ and $\varepsilon$;
2: Sort the hyperedges in $E(v)$ in descending order according to their weights;
3: **for** $i = 1, \cdots, |E(v)|$ **do**
4:     Add all vertices associated with the $i$-th hyperedge to $L$. If $|L| \geq [\frac{1}{\varepsilon}]$, then break;
5: **end for**
6: For each vertex $v_j \in L$, set the corresponding component $x_{v_j}(0) = \frac{1}{|L|}$;
7: **Output:** a prior $x(0)$.

---

For a vertex $v$, the set of hyperedges connected to $v$ is denoted by $E(v)$. We can construct a prior containing $v$ from $E(v)$, which is described in Algorithm 2.

Because of the constraint $x_i \leq \varepsilon$, the initializations need to contain at least $[\frac{1}{\varepsilon}]$ nonzero components. To cover basin of attractions of more maxima, we expect these initializations to locate more uniformly in the space $\{x | x \in \Delta^n, x_i \leq \varepsilon\}$.

Since from every vertex, we can construct such a prior, thus, we can construct $n$ priors in total. From these $n$ priors, according to Algorithm 1, we can obtain $n$ maxima. The significant maxima of (5) are usually among these $n$ maxima, and a significant maximum may appear multiple times. In this way, we can robustly obtain multiple clusters simultaneously, and these clusters may overlap, both of which are desirable properties in many applications. Note that the clustering game approach [12] utilizes a noninformative prior, that is, all vertices have equal probability. Thus, it cannot obtain multiple clusters simultaneously. In clustering game approach [12], if $x_i(t) = 0$, then $x_i(t+1) = 0$, which means that it can only drop points and if a point is initially not included, then it cannot be selected. However, our method can automatically add or drop points, which is another key difference to the clustering game approach.

In each iteration of Algorithm 1, we only need to consider two components of $x$, which makes both the update of rewards and the update of $x(t)$ very efficient. As $f(x(t))$ increases, the sizes of $V_u(x(t))$ and $V_d(x(t))$ both decrease quickly, thus $f(x)$ converges to local maximum quickly. Suppose the maximal number of hyperedges containing a certain vertex is $h$, then the time complexity of Algorithm 1 is $O(thk)$, where $t$ is the number of iterations. The total time complexity of our method is then $O(nthk)$, since we need to ran Algorithm 1 from $n$ initializations.

## 4 Experiments

We evaluate our method on three types of experiments. The first one addresses the problem of line clustering, the second addresses the problem of illumination-invariant face clustering, and the third addresses the problem of affine-invariant point set matching. We compare our method with clique averaging [9] algorithm and matching game approach [12]. In all experiments, the clique averaging approach needs to know the number of clusters in advance; however, both clustering game approach and our method can automatically reveal the number of clusters, which yields the advantages of the latter two in many applications.

### 4.1 Line Clustering

In this experiment, we consider the problem of clustering lines in 2D point sets. Pairwise similarity measures are useless in this case, and at least three points are needed for characterizing such a

property. The dissimilarity measure on triplets of points is given by their mean distance to the best fitting line. If $d(i, j, k)$ is the dissimilarity measure of points $\{i, j, k\}$, then the similarity function is given by $s(\{i, j, k\}) = \exp(-d(i, j, k)^2/\sigma_d^2)$, where $\sigma_d$ is a scaling parameter, which controls the sensitivity of the similarity measure to deformation.

We randomly generate three lines within the region $[-0.5, 0.5]^2$, each line contains 30 points, and all these points have been perturbed by Gaussian noise $N(0, \sigma)$. We also randomly add outliers into the point set. Fig. 1(a) illustrates such a point set with three lines shown in red, blue and green colors, respectively, and the outliers are shown in magenta color. To evaluate the performance, we ran all algorithms on the same data set over 30 trials with varying parameter values, and the performance is measured by F-measure.

We first fix the number of outliers to be 60, vary the scaling parameter $\sigma_d$ from 0.01 to 0.14, and the result is shown in Fig. 1(b). For our method, we set $\varepsilon = 1/30$. Obviously, our method is nearly not affected by the scaling parameter $\sigma_d$, while the clustering game approach is very sensitive to $\sigma_d$. Note that $\sigma_d$ in fact controls the weights of the hyperedge graph and many graph-based algorithms are notoriously sensitive to the weights of the graph. Instead, by setting a proper $\varepsilon$, our method overcomes this problem. From Fig. 1(b), we observe that when $\sigma_d = 4\sigma$, the clustering game approach will get the best performance. Thus, we fix $\sigma_d = 4\sigma$, and change the noise parameter $\sigma$ from 0.01 to 0.1, the results of clustering game approach, clique averaging algorithm and our method are shown in blue, green and red colors in Fig. 1(c), respectively. As the figure shows, when the noise is small, matching game approach outperforms clique averaging algorithm, and when the noise becomes large, the clique averaging algorithm outperforms matching game approach. This is because matching game approach is more robust to outliers, while the clique averaging algorithm seems more robust to noises. Our method always gets the best result, since it can not only select coherent clusters as matching game approach, but also control the size of clusters, thus avoiding the problem of too few points selected into clusters.

In Fig. 1(d) and Fig. 1(e), we vary the number of outliers from 10 to 100, the results clearly demonstrate that our method and clustering game approach are robust to outliers, while clique averaging algorithm is very sensitive to outliers, since it is a partition-based method and every point must be assigned to a cluster. To illustrate the influence of $\varepsilon$, we fix $\sigma_d = \sigma = 0.02$, and test the performance of our method under different $\varepsilon$, the result is shown in Fig. 1(f), note that $x$ axis is $1/\varepsilon$. As we stressed in Section 2, clustering game approach is in fact a special case of our method when $\varepsilon = 1$, thus, the result at $\varepsilon = 1$ is nearly the same as the result of clustering game approach in Fig. 1(b) under the same conditions. Obviously, as $1/\varepsilon$ approaches the real number of points in the cluster, the result become much better. Note that the best result appears when $1/\varepsilon > 30$, which is due to the fact that some outliers fall into the line clusters, as can be seen in Fig. 1(a).

## 4.2 Illumination-invariant face clustering

It has been shown that the variability of images of a Labmertian surface in fixed pose, but under variable lighting conditions where no surface point is shadowed, constitutes a three dimensional linear subspace [15]. This leads to a natural measure of dissimilarity over four images, which can be used for clustering. In fact, this is a generalization of the $k$-lines problem into the $k$-subspaces problem. If we assume that the four images under consideration form the columns of a matrix, and normalize each column by $\ell_2$ norm, then $d = \frac{s_4^2}{s_1^2 + \cdots + s_4^2}$ serves as a natural measure of dissimilarity, where $s_i$ is the $i^{th}$ singular value of this matrix.

In our experiments we use the Yale Face Database B and its extended version [16], which contains 38 individuals, each under 64 different illumination conditions. Since in some lighting conditions, the images are severely shadowed, we delete these images and do the experiments on a subset (about 35 images for each individual). We considered cases where we have faces from 4 and 5 random individuals (randomly choose 10 faces for each individual), with and without outliers. The case with outliers consists 10 additional faces each from a different individual. For each of those combinations, we ran 10 trials to obtain the average F-measures (mean and standard deviation), and the result is reported in Table 1. Note that for each algorithm, we individually tune the parameters to obtain the best results. The results clearly show that partition-based clustering method (clique averaging) is very sensitive to outliers, but performs better when there are no outliers. The clustering game approach and our method both perform well, especially when there are outliers, and our method performs a little better.

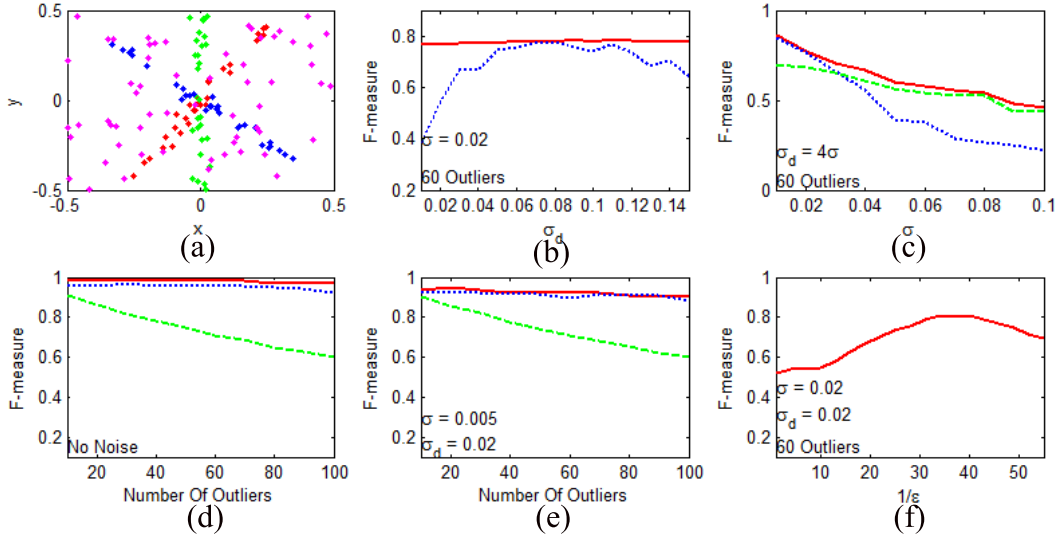

Figure 1: Results on clustering three lines with noises and outliers. The performance of clique averaging algorithm [9], matching game approach [12] and our method is shown as green dashed, blue dotted and read solid curves, respectively. This figure is best viewed in color.

Table 1: Experiments on illuminant-invariant face clustering

| Classes | 4 | | 5 | |
|---|---|---|---|---|
| Outliers | 0 | 10 | 0 | 10 |
| Clique Averaging | **0.95 ± 0.05** | 0.84 ± 0.08 | **0.93 ± 0.05** | 0.83 ± 0.07 |
| Clustering Game | 0.92 ± 0.04 | 0.90 ± 0.04 | 0.91 ± 0.06 | 0.90 ± 0.07 |
| Our Method | 0.93 ± 0.04 | **0.92 ± 0.05** | 0.92 ± 0.07 | **0.91 ± 0.04** |

## 4.3 Affine-invariant Point Set Matching

An important problem in the object recognition is the fact that an object can be seen from different viewpoints, resulting in differently deformed images. Consequently, the invariance to viewpoints is a desirable property for many vision tasks. It is well-known that a near-planar object seen from different viewpoint can be modeled by affine transformations. In this subsection, we will show that matching planar point sets under different viewpoints can be formulated into a hypergraph clustering problem and our algorithm is very suitable for such tasks.

Suppose the two point sets are $P$ and $Q$, with $n_P$ and $n_Q$ points, respectively. For each point in $P$, it may match to any point in $Q$, thus there are $n_P n_Q$ candidate matches. Under the affine transformation $A$, for three correct matches, $m_{ii'}$, $m_{jj'}$ and $m_{kk'}$, $\frac{S_{ijk}}{S_{i'j'k'}} = |\det(A)|$, where $S_{ijk}$ is the area of the triangle formed by points $i$, $j$ and $k$ in $P$, $S_{i'j'k'}$ is the area of the triangle formed by points $i'$, $j'$ and $k'$ in $Q$, and $\det(A)$ is the determinant of $A$. If we regard each candidate match as a point, then $s = \exp(-\frac{(S_{ijk}-S_{i'j'k'}|\det(A)|)^2}{\sigma_d^2})$ serves as a natural similarity measure for three points (candidate matches), $m_{ii'}$, $m_{jj'}$ and $m_{kk'}$, $\sigma_d$ is a scaling parameter, and the correct matching configuration then naturally form a cluster. Note that in this problem, most of the candidate matches are incorrect matches, and can be considered to be outliers.

We did the experiments on 8 shapes from MPEG-7 shape database [17]. For each shape, we uniformly sample its contour into 20 points. Both the shapes and sampled point sets are demonstrated in Fig. 2. We regard original contour point sets as $P$s, then randomly add Gaussian noise $N(0, \sigma)$, and transform them by randomly generated affine matrices $A$s to form corresponding $Q$s. Fig. 3 (a) shows such a pair of $P$ and $Q$ in red and blue, respectively. Since most of points (candidate matches) should not belong to any cluster, partition-based clustering method, such as clique aver-

aging method, cannot be used. Thus, we only compare our method with matching game approach and measure the performance of these two methods by counting how many matches agree with the ground truths. Since $|\det(A)|$ is unknown, we estimate its range and sample several possible values in this range, and conduct the experiment for each possible $|\det(A)|$. In Fig. 3(b), we fix noise parameter $\sigma = 0.05$, and test the robustness of both methods under varying scaling parameter $\sigma_d$. Obviously, our method is very robust to $\sigma_d$, while the matching game approach is very sensitive to it. In Fig. 3(c), we increase $\sigma$ from 0.04 to 0.16, and for each $\sigma$, we adjust $\sigma_d$ to reach the best performances for both methods. As expected, our method is more robust to noise by benefiting from the parameter $\varepsilon$, which is set to 0.05 in both Fig. 3(b) and Fig. 3(c). In Fig. 3(d), we fix $\sigma = 0.05$ and $\sigma_d = 0.15$, and test the performance of our method under different $\varepsilon$. The result again verifies the importance of the parameter $\varepsilon$.

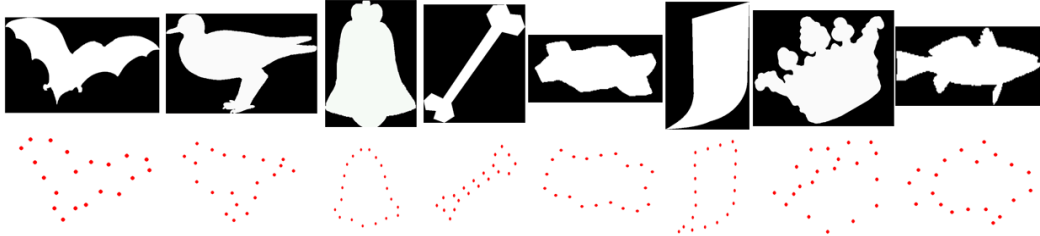

Figure 2: The shapes and corresponding contour point sets used in our experiment.

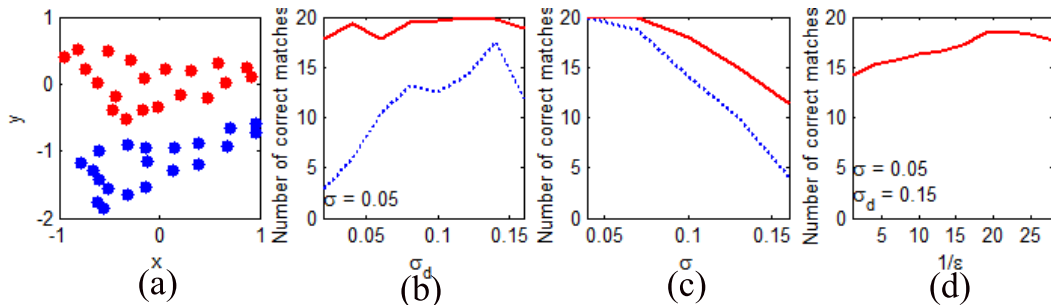

Figure 3: Performance curves on affine-invariant point set matching problem. The red solid curves demonstrate the performance of our method, while the blue dotted curve illustrates the performance of matching game approach.

## 5 Discussion

In this paper, we characterized clustering as an ensemble of all associated affinity relations and relax the clustering problem into optimizing a constrained homogenous function. We showed that the clustering game approach turns out to be a special case of our method. We also proposed an efficient algorithm to automatically reveal the clusters in a data set, even under severe noises and a large number of outliers. The experimental results demonstrated the superiority of our approach with respect to the state-of-the-art counterparts. Especially, our method is not sensitive to the scaling parameter which affects the weights of the graph, and this is a very desirable property in many applications. A key issue with hypergraph-based clustering is the high computational cost of the construction of a hypergraph, and we are currently studying how to efficiently construct an approximate hypergraph and then perform clustering on the incomplete hypergraph.

## 6 Acknowledgement

This research is done for CSIDM Project No. CSIDM-200803 partially funded by a grant from the National Research Foundation (NRF) administered by the Media Development Authority (MDA) of Singapore, and this work has also been partially supported by the NSF Grants IIS-0812118, BCS-0924164 and the AFOSR Grant FA9550-09-1-0207.

# References

[1] A. Jain, M. Murty, and P. Flynn, "Data clustering: a review," *ACM Computing Surveys*, vol. 31, no. 3, pp. 264–323, 1999.

[2] T. Kanungo, D. Mount, N. Netanyahu, C. Piatko, R. Silverman, and A. Wu, "An efficient k-means clustering algorithm: Analysis and implementation," *IEEE Transactions on Pattern Analysis and Machine Intelligence*, vol. 24, no. 7, pp. 881–892, 2002.

[3] A. Ng, M. Jordan, and Y. Weiss, "On spectral clustering: Analysis and an algorithm," in *Advances in Neural Information Processing Systems*, vol. 2, 2002, pp. 849–856.

[4] I. Dhillon, Y. Guan, and B. Kulis, "Kernel k-means: spectral clustering and normalized cuts," in *Proceedings of the tenth ACM International Conference on Knowledge Discovery and Data Mining*, 2004, pp. 551–556.

[5] J. Shi and J. Malik, "Normalized cuts and image segmentation," *IEEE Transactions on Pattern Analysis and Machine Intelligence*, vol. 22, no. 8, pp. 888–905, 2000.

[6] B. Frey and D. Dueck, "Clustering by passing messages between data points," *Science*, vol. 315, no. 5814, pp. 972–976, 2007.

[7] J. Zien, M. Schlag, and P. Chan, "Multilevel spectral hypergraph partitioning with arbitrary vertex sizes," *IEEE Transactions on Computer-aided Design of Integrated Circuits and Systems*, vol. 18, no. 9, pp. 1389–1399, 1999.

[8] J. Rodriguez, "On the Laplacian spectrum and walk-regular hypergraphs," *Linear and Multilinear Algebra*, vol. 51, no. 3, pp. 285–297, 2003.

[9] S. Agarwal, J. Lim, L. Zelnik-Manor, P. Perona, D. Kriegman, and S. Belongie, "Beyond pairwise clustering," in *IEEE Computer Society Conference on Computer Vision and Pattern Recognition*, vol. 2, 2005, pp. 838–845.

[10] D. Zhou, J. Huang, and B. Scholkopf, "Learning with hypergraphs: Clustering, classification, and embedding," in *Advances in Neural Information Processing Systems*, vol. 19, 2007, pp. 1601–1608.

[11] A. Shashua, R. Zass, and T. Hazan, "Multi-way clustering using super-symmetric non-negative tensor factorization," in *European Conference on Computer Vision*, 2006, pp. 595–608.

[12] S. Bulo and M. Pelillo, "A game-theoretic approach to hypergraph clustering," in *Advances in Neural Information Processing Systems*, 2009.

[13] M. Pavan and M. Pelillo, "Dominant sets and pairwise clustering," *IEEE Transactions on Pattern Analysis and Machine Intelligence*, vol. 29, no. 1, pp. 167–172, 2007.

[14] H. Kuhn and A. Tucker, "Nonlinear programming," *ACM SIGMAP Bulletin*, pp. 6–18, 1982.

[15] P. Belhumeur and D. Kriegman, "What is the set of images of an object under all possible illumination conditions?" *International Journal of Computer Vision*, vol. 28, no. 3, pp. 245–260, 1998.

[16] K. Lee, J. Ho, and D. Kriegman, "Acquiring linear subspaces for face recognition under variable lighting," *IEEE Transactions on Pattern Analysis and Machine Intelligence*, vol. 27, no. 5, pp. 684–698, 2005.

[17] L. Latecki, R. Lakamper, and T. Eckhardt, "Shape descriptors for non-rigid shapes with a single closed contour," in *IEEE Conference on Computer Vision and Pattern Recognition*, vol. 1, 2000, pp. 65–72.

